# The Stability of Kernel Principal Components Analysis and its Relation to the Process Eigenspectrum

**John Shawe-Taylor**
Royal Holloway
University of London
john@cs.rhul.ac.uk

**Christopher K. I. Williams**
School of Informatics
University of Edinburgh
c.k.i.williams@ed.ac.uk

## Abstract

In this paper we analyze the relationships between the eigenvalues of the $m \times m$ Gram matrix $K$ for a kernel $k(\cdot, \cdot)$ corresponding to a sample $\mathbf{x}_1, \ldots, \mathbf{x}_m$ drawn from a density $p(\mathbf{x})$ and the eigenvalues of the corresponding continuous eigenproblem. We bound the differences between the two spectra and provide a performance bound on kernel PCA.

## 1 Introduction

Over recent years there has been a considerable amount of interest in kernel methods for supervised learning (e.g. Support Vector Machines and Gaussian Process prediction) and for unsupervised learning (e.g. kernel PCA, Schölkopf et al. (1998)). In this paper we study the stability of the subspace of feature space extracted by kernel PCA with respect to the sample of size $m$, and relate this to the feature space that would be extracted in the infinite sample-size limit. This analysis essentially "lifts" into (a potentially infinite dimensional) feature space an analysis which can also be carried out for PCA, comparing the $k$-dimensional eigenspace extracted from a sample covariance matrix and the $k$-dimensional eigenspace extracted from the population covariance matrix, and comparing the residuals from the $k$-dimensional compression for the $m$-sample and the population.

Earlier work by Shawe-Taylor et al. (2002) discussed the concentration of spectral properties of Gram matrices and of the residuals of fixed projections. However, these results gave deviation bounds on the sampling variability the eigenvalues of the Gram matrix, but did not address the relationship of sample and population eigenvalues, or the estimation problem of the residual of PCA on new data.

The structure the remainder of the paper is as follows. In section 2 we provide background on the continuous kernel eigenproblem, and the relationship between the eigenvalues of certain matrices and the expected residuals when projecting into spaces of dimension $k$. Section 3 provides inequality relationships between the process eigenvalues and the expectation of the Gram matrix eigenvalues. Section 4 presents some concentration results and uses these to develop an approximate chain of inequalities. In section 5 we obtain a performance bound on kernel PCA, relating the performance on the training sample to the expected performance wrt $p(\mathbf{x})$.

## 2 Background

### 2.1 The kernel eigenproblem

For a given kernel function $k(\cdot, \cdot)$ the $m \times m$ Gram matrix $K$ has entries $k(\mathbf{x}_i, \mathbf{x}_j)$, $i, j = 1, \ldots, m$, where $\{\mathbf{x}_i : i = 1, \ldots, m\}$ is a given dataset. For Mercer kernels $K$ is symmetric positive semi-definite. We denote the eigenvalues of the Gram matrix as $\hat{\lambda}_1 \geq \hat{\lambda}_2 \ldots \geq \hat{\lambda}_m \geq 0$ and write its eigendecomposition as $K = Z\hat{\Lambda}Z'$ where $\hat{\Lambda}$ is a diagonal matrix of the eigenvalues and $Z'$ denotes the transpose of matrix $Z$. The eigenvalues are also referred to as the spectrum of the Gram matrix.

We now describe the relationship between the eigenvalues of the Gram matrix and those of the underlying process. For a given kernel function and density $p(\mathbf{x})$ on a space $\mathcal{X}$, we can also write down the eigenfunction problem

$$\int_{\mathcal{X}} k(\mathbf{x}, \mathbf{y})p(\mathbf{x})\phi_i(\mathbf{x}) \, d\mathbf{x} = \lambda_i \phi_i(\mathbf{y}). \tag{1}$$

Note that the eigenfunctions are orthonormal with respect to $p(\mathbf{x})$, i.e. $\int_{\mathcal{X}} \phi_i(\mathbf{x})p(\mathbf{x})\phi_j(\mathbf{x})d\mathbf{x} = \delta_{ij}$. Let the eigenvalues be ordered so that $\lambda_1 \geq \lambda_2 \geq \ldots$. This continuous eigenproblem can be approximated in the following way. Let $\{\mathbf{x}_i : i = 1, \ldots, m\}$ be a sample drawn according to $p(\mathbf{x})$. Then as pointed out in Williams and Seeger (2000), we can approximate the integral with weight function $p(\mathbf{x})$ by an average over the sample points, and then plug in $\mathbf{y} = \mathbf{x}_j$ for $j = 1, \ldots, m$ to obtain the matrix eigenproblem.

$$\frac{1}{m}\sum_{k=1}^{m} k(\mathbf{x}_k, \mathbf{x}_j)\phi_i(\mathbf{x}_k) = \lambda_i \phi_i(\mathbf{x}_j).$$

Thus we see that $\mu_i \overset{def}{=} \frac{1}{m}\hat{\lambda}_i$ is an obvious estimator for the $i$th eigenvalue of the continuous problem. The theory of the numerical solution of eigenvalue problems (Baker 1977, Theorem 3.4) shows that for a fixed $k$, $\mu_k$ will converge to $\lambda_k$ in the limit as $m \to \infty$.

For the case that $\mathcal{X}$ is one dimensional, $p(x)$ is Gaussian and $k(x, y) = \exp -b(x - y)^2$, there are analytic results for the eigenvalues and eigenfunctions of equation (1) as given in section 4 of Zhu et al. (1998). A plot in Williams and Seeger (2000) for $m = 500$ with $b = 3$ and $p(x) \sim N(0, 1/4)$ shows good agreement between $\mu_i$ and $\lambda_i$ for small $i$, but that for larger $i$ the matrix eigenvalues underestimate the process eigenvalues. One of the by-products of this paper will be bounds on the degree of underestimation for this estimation problem in a fully general setting.

Koltchinskii and Gine (2000) discuss a number of results including rates of convergence of the $\mu$-spectrum to the $\lambda$-spectrum. The measure they use compares the whole spectrum rather than individual eigenvalues or subsets of eigenvalues. They also do not deal with the estimation problem for PCA residuals.

### 2.2 Projections, residuals and eigenvalues

The approach adopted in the proofs of the next section is to relate the eigenvalues to the sums of squares of residuals. Let $\mathbf{x}$ be a random variable in $d$ dimensions, and let $X$ be a $d \times m$ matrix containing $m$ sample vectors $\mathbf{x}_1, \ldots, \mathbf{x}_m$. Consider the $m \times m$ matrix $M = X'X$ with eigendecomposition $M = Z\hat{\Lambda}Z'$. Then taking $X = Z\sqrt{\hat{\Lambda}}$ we obtain a finite dimensional version of Mercer's theorem. To set the scene, we now present a short description of the residuals viewpoint.

The starting point is the singular value decomposition of $X = U\Sigma Z'$, where $U$ and $Z$ are orthonormal matrices and $\Sigma$ is a diagonal matrix containing the singular

values (in descending order). We can now reconstruct the eigenvalue decomposition of $M = X'X = Z\Sigma U'U\Sigma Z' = Z\hat{\Lambda}Z'$, where $\hat{\Lambda} = \Sigma^2$. But equally we can construct a $d \times d$ matrix $N = XX' = U\Sigma Z'Z\Sigma U' = U\hat{\Lambda}U'$, with the same eigenvalues as $M$. We have made a slight abuse of notation by using $\hat{\Lambda}$ to represent two matrices of potentially different dimensions, but the larger is simply an extension of the smaller with 0's. Note that $N = mC_X$, where $C_X$ is the sample correlation matrix.

Let $V$ be a linear space spanned by $k$ linearly independent vectors. Let $P_V(\mathbf{x})$ ($P_V^\perp(\mathbf{x})$) be the projection of $\mathbf{x}$ onto $V$ (space perpendicular to $V$), so that $\|\mathbf{x}\|^2 = \|P_V(\mathbf{x})\|^2 + \|P_V^\perp(\mathbf{x})\|^2$. Using the Courant-Fisher minimax theorem it can be proved (Shawe-Taylor et al., 2002, equation 4) that

$$\sum_{i=1}^{k} \hat{\lambda}_i(M) = \max_{\dim(V)=k} \sum_{j=1}^{m} \|P_V(\mathbf{x}_j)\|^2 = \sum_{j=1}^{m} \|\mathbf{x}_j\|^2 - \min_{\dim(V)=k} \sum_{j=1}^{m} \|P_V^\perp(\mathbf{x}_j)\|^2,$$

$$\sum_{i=k+1}^{m} \hat{\lambda}_i(M) = \sum_{j=1}^{m} \|\mathbf{x}_j\|^2 - \sum_{i=1}^{k} \hat{\lambda}_i(M) = \min_{\dim(V)=k} \sum_{j=1}^{m} \|P_V^\perp(\mathbf{x}_j)\|^2. \quad (2)$$

Hence the subspace spanned by the first $k$ eigenvectors is characterised as that for which the sum of the squares of the residuals is minimal. We can also obtain similar results for the population case, e.g. $\sum_{i=1}^{k} \lambda_i = \max_{\dim(V)=k} \mathbb{E}[\|P_V(\mathbf{x})\|^2]$.

## 2.3   Residuals in feature space

Frequently, we consider all of the above as occurring in a kernel defined feature space, so that wherever we have written a vector $\mathbf{x}$ we should have put $\psi(\mathbf{x})$, where $\psi$ is the corresponding feature map $\psi : \mathbf{x} \in \mathcal{X} \longmapsto \psi(\mathbf{x}) \in F$ to a feature space $F$. Hence, the matrix $M$ has entries $M_{ij} = \langle \psi(\mathbf{x}_i), \psi(\mathbf{x}_j) \rangle$. The kernel function computes the composition of the inner product with the feature maps, $k(\mathbf{x}, \mathbf{z}) = \langle \psi(\mathbf{x}), \psi(\mathbf{z}) \rangle = \psi(\mathbf{x})'\psi(\mathbf{z})$, which can in many cases be computed without explicitly evaluating the mapping $\psi$. We would also like to evaluate the projections into eigenspaces without explicitly computing the feature mapping $\psi$. This can be done as follows. Let $\mathbf{u}_i$ be the $i$-th singular vector in the feature space, that is the $i$-th eigenvector of the matrix $N$, with the corresponding singular value being $\sigma_i = \sqrt{\hat{\lambda}_i}$ and the corresponding eigenvector of $M$ being $\mathbf{z}_i$. The projection of an input $\mathbf{x}$ onto $\mathbf{u}_i$ is given by

$$\psi(\mathbf{x})'\mathbf{u}_i = (\psi(\mathbf{x})'U)_i = (\psi(\mathbf{x})'XZ)_i\sigma_i^{-1} = \mathbf{k}'\mathbf{z}_i\sigma_i^{-1},$$

where we have used the fact that $X = U\Sigma Z'$ and $\mathbf{k}_j = \psi(\mathbf{x})'\psi(\mathbf{x}_j) = k(\mathbf{x}, \mathbf{x}_j)$.

Our final background observation concerns the kernel operator and its eigenspaces. The operator in question is

$$K(f)(\mathbf{x}) = \int_{\mathcal{X}} k(\mathbf{x}, \mathbf{z})f(\mathbf{z})p(\mathbf{z})d\mathbf{z}.$$

Provided the operator is positive semi-definite, by Mercer's theorem we can decompose $k(\mathbf{x}, \mathbf{z})$ as a sum of eigenfunctions, $k(\mathbf{x}, \mathbf{z}) = \sum_{i=1}^{\infty} \lambda_i \phi_i(\mathbf{x})\phi_i(\mathbf{z}) = \langle \psi(\mathbf{x}), \psi(\mathbf{z}) \rangle$, where the functions $(\phi_i(\mathbf{x}))_{i=1}^{\infty}$ form a complete orthonormal basis with respect to the inner product $\langle f, g \rangle_p = \int_{\mathcal{X}} f(\mathbf{x})g(\mathbf{x})p(\mathbf{x})dx$ and $\psi(\mathbf{x})$ is the feature space mapping

$$\psi : \mathbf{x} \longrightarrow (\psi_i(\mathbf{x}))_{i=1}^{\infty} = \left(\sqrt{\lambda_i}\phi_i(\mathbf{x})\right)_{i=1}^{\infty} \in F.$$

Note that $\phi_i(\mathbf{x})$ has norm 1 and satisfies $\lambda_i \phi_i(\mathbf{x}) = \int_{\mathcal{X}} k(\mathbf{x}, \mathbf{z})\phi_i(\mathbf{z})p(\mathbf{z})d\mathbf{z}$ (equation 1), so that

$$\lambda_i = \int_{\mathcal{X}^2} k(\mathbf{y}, \mathbf{z})\phi_i(\mathbf{y})\phi_i(\mathbf{z})p(\mathbf{z})p(\mathbf{y})d\mathbf{y}d\mathbf{z}. \quad (3)$$

If we let $\boldsymbol{\phi}(\mathbf{x}) = (\phi_i(\mathbf{x}))_{i=1}^{\infty} \in F$, we can define the unit vector $\mathbf{u}_i \in F$ corresponding to $\lambda_i$ by $\mathbf{u}_i = \int_{\mathcal{X}} \phi_i(\mathbf{x})\boldsymbol{\phi}(\mathbf{x})p(\mathbf{x})d\mathbf{x}$. For a general function $f(\mathbf{x})$ we can similarly define the vector $\mathbf{f} = \int_{\mathcal{X}} f(\mathbf{x})\boldsymbol{\phi}(\mathbf{x})p(\mathbf{x})d\mathbf{x}$. Now the expected square of the norm of the projection $P_{\mathbf{f}}(\boldsymbol{\psi}(\mathbf{x}))$ onto the vector $\mathbf{f}$ (assumed to be of norm 1) of an input $\boldsymbol{\psi}(\mathbf{x})$ drawn according to $p(\mathbf{x})$ is given by

$$
\begin{aligned}
\mathbb{E}\left[\|P_{\mathbf{f}}(\boldsymbol{\psi}(\mathbf{x}))\|^2\right] &= \int_{\mathcal{X}} \|P_{\mathbf{f}}(\boldsymbol{\psi}(\mathbf{x}))\|^2 p(\mathbf{x})d\mathbf{x} = \int_{\mathcal{X}} \left(\mathbf{f}'\boldsymbol{\psi}(\mathbf{x})\right)^2 p(\mathbf{x})d\mathbf{x} \\
&= \int_{\mathcal{X}}\int_{\mathcal{X}}\int_{\mathcal{X}} f(\mathbf{y})\boldsymbol{\phi}(\mathbf{y})'\boldsymbol{\psi}(\mathbf{x})p(\mathbf{y})d\mathbf{y} f(\mathbf{z})\boldsymbol{\phi}(\mathbf{z})'\boldsymbol{\psi}(\mathbf{x})p(\mathbf{z})d\mathbf{z}p(\mathbf{x})d\mathbf{x} \\
&= \int_{\mathcal{X}^3} f(\mathbf{y})f(\mathbf{z}) \sum_{j=1}^{\infty} \sqrt{\lambda_j}\phi_j(\mathbf{y})\phi_j(\mathbf{x})p(\mathbf{y})d\mathbf{y} \sum_{\ell=1}^{\infty} \sqrt{\lambda_\ell}\phi_\ell(\mathbf{z})\phi_\ell(\mathbf{x})p(\mathbf{z})d\mathbf{z}p(\mathbf{x})d\mathbf{x} \\
&= \int_{\mathcal{X}^2} f(\mathbf{y})f(\mathbf{z}) \sum_{j,\ell=1}^{\infty} \sqrt{\lambda_j}\phi_j(\mathbf{y})p(\mathbf{y})d\mathbf{y}\sqrt{\lambda_\ell}\phi_\ell(\mathbf{z})p(\mathbf{z})d\mathbf{z} \int_{\mathcal{X}} \phi_j(\mathbf{x})\phi_\ell(\mathbf{x})p(\mathbf{x})d\mathbf{x} \\
&= \int_{\mathcal{X}^2} f(\mathbf{y})f(\mathbf{z}) \sum_{j=1}^{\infty} \lambda_j\phi_j(\mathbf{y})\phi_j(\mathbf{z})p(\mathbf{y})d\mathbf{y}p(\mathbf{z})d\mathbf{z} \\
&= \int_{\mathcal{X}^2} f(\mathbf{y})f(\mathbf{z})k(\mathbf{y},\mathbf{z})p(\mathbf{y})p(\mathbf{z})d\mathbf{y}d\mathbf{z}.
\end{aligned}
$$

Since all vectors $\mathbf{f}$ in the subspace spanned by the image of the input space in $F$ can be expressed in this fashion, it follows using (3) that the sum of the finite case characterisation of eigenvalues and eigenvectors is replaced by an expectation

$$
\lambda_k = \max_{\dim(V)=k} \min_{0 \neq \mathbf{v} \in V} \mathbb{E}[\|P_{\mathbf{v}}(\boldsymbol{\psi}(\mathbf{x}))\|^2], \tag{4}
$$

where $V$ is a linear subspace of the feature space $F$. Similarly,

$$
\sum_{i=1}^{k} \lambda_i = \max_{\dim(V)=k} \mathbb{E}\left[\|P_V(\boldsymbol{\psi}(\mathbf{x}))\|^2\right] = \mathbb{E}\left[\|\boldsymbol{\psi}(\mathbf{x})\|^2\right] - \min_{\dim(V)=k} \mathbb{E}\left[\|P_V^\perp(\boldsymbol{\psi}(\mathbf{x}))\|^2\right],
$$

$$
\sum_{i=k+1}^{\infty} \lambda_i = \mathbb{E}\left[\|\boldsymbol{\psi}(\mathbf{x})\|^2\right] - \sum_{i=1}^{k} \lambda_i = \min_{\dim(V)=k} \mathbb{E}\left[\|P_V^\perp(\boldsymbol{\psi}(\mathbf{x}))\|^2\right]. \tag{5}
$$

where $P_V(\boldsymbol{\psi}(\mathbf{x}))$ $(P_V^\perp(\boldsymbol{\psi}(\mathbf{x})))$ is the projection of $\boldsymbol{\psi}(\mathbf{x})$ into the subspace $V$ (the projection of $\boldsymbol{\psi}(\mathbf{x})$ into the space orthogonal to $V$).

### 2.4   Plan of campaign

We are now in a position to motivate the main results of the paper. We consider the general case of a kernel defined feature space with input space $\mathcal{X}$ and probability density $p(\mathbf{x})$. We fix a sample size $m$ and a draw of $m$ examples $S = (\mathbf{x}_1, \mathbf{x}_2, \ldots, \mathbf{x}_m)$ according to $p$. Further we fix a feature dimension $k$. Let $\hat{V}_k$ be the space spanned by the first $k$ eigenvectors of the sample kernel matrix $K$ with corresponding eigenvalues $\hat{\lambda}_1, \hat{\lambda}_2, \ldots, \hat{\lambda}_k$, while $V_k$ is the space spanned by the first $k$ process eigenvectors with corresponding eigenvalues $\lambda_1, \lambda_2, \ldots, \lambda_k$. Similarly, let $\hat{\mathbb{E}}[f(\mathbf{x})]$ denote expectation with respect to the sample, $\hat{\mathbb{E}}[f(\mathbf{x})] = \frac{1}{m}\sum_{i=1}^{m} f(\mathbf{x}_i)$, while as before $\mathbb{E}[\cdot]$ denotes expectation with respect to $p$.

We are interested in the relationships between the following quantities: (i) $\hat{\mathbb{E}}\left[\|P_{\hat{V}_k}(\mathbf{x})\|^2\right] = \frac{1}{m}\sum_{i=1}^{k} \hat{\lambda}_i = \sum_{i=1}^{k} \mu_i$, (ii) $\mathbb{E}\left[\|P_{V_k}(\mathbf{x})\|^2\right] = \sum_{i=1}^{k} \lambda_i$ (iii)

$\mathbb{E}\left[\|P_{\hat{V}_k}(\mathbf{x})\|^2\right]$ and (iv) $\hat{\mathbb{E}}\left[\|P_{V_k}(\mathbf{x})\|^2\right]$. Bounding the difference between the first and second will relate the process eigenvalues to the sample eigenvalues, while the difference between the first and third will bound the expected performance of the space identified by kernel PCA when used on new data.

Our first two observations follow simply from equation (5),

$$\hat{\mathbb{E}}\left[\|P_{\hat{V}_k}(\mathbf{x})\|^2\right] \; = \; \frac{1}{m}\sum_{i=1}^{k}\hat{\lambda}_i \geq \hat{\mathbb{E}}\left[\|P_{V_k}(\mathbf{x})\|^2\right], \tag{6}$$

$$\text{and} \quad \mathbb{E}\left[\|P_{V_k}(\mathbf{x})\|^2\right] \; = \; \sum_{i=1}^{k}\lambda_i \geq \mathbb{E}\left[\|P_{\hat{V}_k}(\mathbf{x})\|^2\right]. \tag{7}$$

Our strategy will be to show that the right hand side of inequality (6) and the left hand side of inequality (7) are close in value making the two inequalities approximately a chain of inequalities. We then bound the difference between the first and last entries in the chain.

## 3  Averaging over Samples and Population Eigenvalues

The sample correlation matrix is $C_X = \frac{1}{m}XX'$ with eigenvalues $\mu_1 \geq \mu_2 \ldots \geq \mu_d$. In the notation of the section 2 $\mu_i = (1/m)\hat{\lambda}_i$. The corresponding population correlation matrix has eigenvalues $\lambda_1 \geq \lambda_2 \ldots \geq \lambda_d$ and eigenvectors $\mathbf{u}_1, \ldots, \mathbf{u}_d$. Again by the observations above these are the process eigenvalues. Let $\mathbb{E}_m[\cdot]$ denote averages over random samples of size $m$.

The following proposition describes how $\mathbb{E}_m[\mu_1]$ is related to $\lambda_1$ and $\mathbb{E}_m[\mu_d]$ is related to $\lambda_d$. It requires no assumption of Gaussianity.

**Proposition 1 (Anderson, 1963, pp 145-146)** $\mathbb{E}_m[\mu_1] \geq \lambda_1$ and $\mathbb{E}_m[\mu_d] \leq \lambda_d$.

**Proof**: By the results of the previous section we have

$$\mu_1 = \max_{0 \neq \mathbf{c}} \sum_{i=1}^{m}\frac{1}{m}\|P_{\mathbf{c}}(\mathbf{x}_i)\|^2 \geq \frac{1}{m}\sum_{i=1}^{m}\|P_{\mathbf{u}_1}(\mathbf{x}_i)\|^2 = \hat{\mathbb{E}}\left[\|P_{\mathbf{u}_1}(\mathbf{x})\|^2\right].$$

We now apply the expectation operator $\mathbb{E}_m$ to both sides. On the RHS we get

$$\mathbb{E}_m\hat{\mathbb{E}}\left[\|P_{\mathbf{u}_1}(\mathbf{x})\|^2\right] = \mathbb{E}\left[\|P_{\mathbf{u}_1}(\mathbf{x})\|^2\right] = \lambda_1$$

by equation (5), which completes the proof. Correspondingly $\mu_d$ is characterized by $\mu_d = \min_{0 \neq \mathbf{c}} \hat{\mathbb{E}}\left[\|P_{\mathbf{c}}(\mathbf{x}_i)\|^2\right]$ (minor components analysis). □

Interpreting this result, we see that $\mathbb{E}_m[\mu_1]$ *overestimates* $\lambda_1$, while $\mathbb{E}_m[\mu_d]$ *underestimates* $\lambda_d$.

Proposition 1 can be generalized to give the following result where we have also allowed for a kernel defined feature space of dimension $N_F \leq \infty$.

**Proposition 2** *Using the above notation, for any $k$, $1 \leq k \leq m$, $\mathbb{E}_m[\sum_{i=1}^{k}\mu_i] \geq \sum_{i=1}^{k}\lambda_i$ and $\mathbb{E}_m[\sum_{i=k+1}^{m}\mu_i] \leq \sum_{i=k+1}^{N_F}\lambda_i$.*

**Proof**: Let $V_k$ be the space spanned by the first $k$ process eigenvectors. Then from the derivations above we have

$$\sum_{i=1}^{k}\mu_i = \max_{V:\,\dim V = k} \hat{\mathbb{E}}\left[\|P_V(\psi(\mathbf{x}))\|^2\right] \geq \hat{\mathbb{E}}\left[\|P_{V_k}(\psi(\mathbf{x}))\|^2\right].$$

Again, applying the expectation operator $\mathbb{E}_m$ to both sides of this equation and taking equation (5) into account, the first inequality follows. To prove the second we turn max into min, $P$ into $P^\perp$ and reverse the inequality. Again taking expectations of both sides proves the second part. $\square$

Applying the results obtained in this section, it follows that $\mathbb{E}_m[\mu_1]$ will overestimate $\lambda_1$, and the cumulative sum $\sum_{i=1}^k \mathbb{E}_m[\mu_i]$ will overestimate $\sum_{i=1}^k \lambda_i$. At the other end, clearly for $N_F \geq k > m$, $\mu_k \equiv 0$ is an underestimate of $\lambda_k$.

## 4    Concentration of eigenvalues

We now make use of results from Shawe-Taylor et al. (2002) concerning the concentration of the eigenvalue spectrum of the Gram matrix. We have

**Theorem 3** *Let $K(\mathbf{x}, \mathbf{z})$ be a positive semi-definite kernel function on a space $X$, and let $p$ be a probability density function on $X$. Fix natural numbers $m$ and $1 \leq k < m$ and let $S = (\mathbf{x}_1, \ldots, \mathbf{x}_m) \in X^m$ be a sample of $m$ points drawn according to $p$. Then for all $\epsilon > 0$,*

$$P\left\{\left|\frac{1}{m}\hat{\lambda}^{\leq k}(S) - \mathbb{E}_m\left[\frac{1}{m}\hat{\lambda}^{\leq k}(S)\right]\right| \geq \epsilon\right\} \leq 2\exp\left(\frac{-2\epsilon^2 m}{R^4}\right),$$

*where $\hat{\lambda}^{\leq k}(S)$ is the sum of the largest $k$ eigenvalues of the matrix $K(S)$ with entries $K(S)_{ij} = K(\mathbf{x}_i, \mathbf{x}_j)$ and $R^2 = \max_{\mathbf{x} \in X} K(\mathbf{x}, \mathbf{x})$.*

This follows by a similar derivation to Theorem 5 in Shawe-Taylor et al. (2002).

Our next result concerns the concentration of the residuals with respect to a fixed subspace. For a subspace $V$ and training set $S$, we introduce the notation

$$\bar{P}_V(S) = \hat{\mathbb{E}}\left[\|P_V(\boldsymbol{\psi}(\mathbf{x}))\|^2\right].$$

**Theorem 4** *Let $p$ be a probability density function on $X$. Fix natural numbers $m$ and a subspace $V$ and let $S = (\mathbf{x}_1, \ldots, \mathbf{x}_m) \in X^m$ be a sample of $m$ points drawn according to a probability density function $p$. Then for all $\epsilon > 0$,*

$$P\{\bar{P}_V(S) - \mathbb{E}_m[\bar{P}_V(S)]| \geq \epsilon\} \leq 2\exp\left(\frac{-\epsilon^2 m}{2R^4}\right).$$

This is theorem 6 in Shawe-Taylor et al. (2002).

The concentration results of this section are very tight. In the notation of the earlier sections they show that with high probability

$$\hat{\mathbb{E}}\left[\|P_{\hat{V}_k}(\boldsymbol{\psi}(\mathbf{x}))\|^2\right] = \frac{1}{m}\sum_{i=1}^k \hat{\lambda}_i \approx \mathbb{E}_m\left[\hat{\mathbb{E}}\left[\|P_{\hat{V}_k}(\boldsymbol{\psi}(\mathbf{x}))\|^2\right]\right] = \mathbb{E}_m\left[\frac{1}{m}\sum_{i=1}^k \hat{\lambda}_i\right] \quad (8)$$

and

$$\mathbb{E}\left[\|P_{V_k}(\boldsymbol{\psi}(\mathbf{x}))\|^2\right] = \sum_{i=1}^k \lambda_i \approx \hat{\mathbb{E}}\left[\|P_{V_k}(\boldsymbol{\psi}(\mathbf{x}))\|^2\right], \quad (9)$$

where we have used Theorem 3 to obtain the first approximate equality and Theorem 4 with $V = V_k$ to obtain the second approximate equality.

This gives the sought relationship to create an approximate chain of inequalities

$$\hat{\mathbb{E}}\left[\|P_{\hat{V}_k}(\boldsymbol{\psi}(\mathbf{x}))\|^2\right] = \frac{1}{m}\sum_{i=1}^k \hat{\lambda}_i \geq \hat{\mathbb{E}}\left[\|P_{V_k}(\boldsymbol{\psi}(\mathbf{x}))\|^2\right]$$

$$\approx \mathbb{E}\left[\|P_{V_k}(\boldsymbol{\psi}(\mathbf{x}))\|^2\right] = \sum_{i=1}^k \lambda_i \geq \mathbb{E}\left[\|P_{\hat{V}_k}(\boldsymbol{\psi}(\mathbf{x}))\|^2\right]. \quad (10)$$

This approximate chain of inequalities could also have been obtained using Proposition 2. It remains to bound the difference between the first and last entries in this chain. This together with the concentration results of this section will deliver the required bounds on the differences between empirical and process eigenvalues, as well as providing a performance bound on kernel PCA.

## 5   Learning a projection matrix

The key observation that enables the analysis bounding the difference between $\hat{\mathbb{E}}\left[\|P_{\hat{V}_k}(\psi(\mathbf{x}))\|^2\right]$ and $\mathbb{E}\left[\|P_{\hat{V}_k}(\psi(\mathbf{x}))\|^2\right]$ is that we can view the projection norm $\|P_{\hat{V}_k}(\psi(\mathbf{x}))\|^2$ as a linear function of pairs of features from the feature space $F$.

**Proposition 5** *The projection norm $\|P_{\hat{V}_k}(\psi(\mathbf{x}))\|^2$ is a linear function $\hat{f}$ in a feature space $\hat{F}$ for which the kernel function is given by $\hat{k}(\mathbf{x},\mathbf{z}) = k(\mathbf{x},\mathbf{z})^2$. Furthermore the 2-norm of the function $\hat{f}$ is $\sqrt{k}$.*

**Proof**: Let $X = U\Sigma Z'$ be the singular value decomposition of the sample matrix $X$ in the feature space. The projection norm is then given by $\hat{f}(\mathbf{x}) = \|P_{\hat{V}_k}(\psi(\mathbf{x}))\|^2 = \psi(\mathbf{x})'U_kU_k'\psi(\mathbf{x})$, where $U_k$ is the matrix containing the first $k$ columns of $U$. Hence we can write

$$\|P_{\hat{V}_k}(\psi(\mathbf{x}))\|^2 = \sum_{ij=1}^{N_F} \alpha_{ij}\psi(\mathbf{x})_i\psi(\mathbf{x})_j = \sum_{ij=1}^{N_F} \alpha_{ij}\hat{\psi}(\mathbf{x})_{ij},$$

where $\hat{\psi}$ is the projection mapping into the feature space $\hat{F}$ consisting of all pairs of $F$ features and $\alpha_{ij} = (U_kU_k')_{ij}$. The standard polynomial construction gives

$$
\begin{aligned}
\hat{k}(\mathbf{x},\mathbf{z}) &= k(\mathbf{x},\mathbf{z})^2 = \left(\sum_{i=1}^{N_F}\psi(\mathbf{x})_i\psi(\mathbf{z})_i\right)^2 \\
&= \sum_{i,j=1}^{N_F}\psi(\mathbf{x})_i\psi(\mathbf{z})_i\psi(\mathbf{x})_j\psi(\mathbf{z})_j = \sum_{i,j=1}^{N_F}(\psi(\mathbf{x})_i\psi(\mathbf{x})_j)(\psi(\mathbf{z})_i\psi(\mathbf{z})_j) \\
&= \left\langle \hat{\psi}(\mathbf{x}),\hat{\psi}(\mathbf{z})\right\rangle_{\hat{F}}.
\end{aligned}
$$

It remains to show that the norm of the linear function is $k$. The norm satisfies (note that $\|\cdot\|_F$ denotes the Frobenius norm and $\mathbf{u}_i$ the columns of $U$)

$$\|\hat{f}\|^2 = \sum_{i,j=1}^{N_F}\alpha_{ij}^2 = \|U_kU_k'\|_F^2 = \left\langle\sum_{i=1}^k \mathbf{u}_i\mathbf{u}_i', \sum_{j=1}^k \mathbf{u}_j\mathbf{u}_j'\right\rangle_F = \sum_{i,j=1}^k (\mathbf{u}_i'\mathbf{u}_j)^2 = k$$

as required. $\square$

We are now in a position to apply a learning theory bound where we consider a regression problem for which the target output is the square of the norm of the sample point $\|\psi(\mathbf{x})\|^2$. We restrict the linear function in the space $\hat{F}$ to have norm $\sqrt{k}$. The loss function is then the shortfall between the output of $\hat{f}$ and the squared norm.

Using Rademacher complexity theory we can obtain the following theorems:

**Theorem 6** *If we perform PCA in the feature space defined by a kernel $k(\mathbf{x},\mathbf{z})$ then with probability greater than $1-\delta$, for all $1 \leq k \leq m$, if we project new data*

onto the space $\hat{V}_k$, the expected squared residual is bounded by

$$\lambda^{>k} \leq \mathbb{E}\left[\|P_{\hat{V}_k}^{\perp}(\psi(\mathbf{x}))\|^2\right] \leq \min_{1 \leq \ell \leq k}\left[\frac{1}{m}\hat{\lambda}^{>\ell}(S) + \frac{1+\sqrt{\ell}}{\sqrt{m}}\sqrt{\frac{2}{m}\sum_{i=1}^{m}k(\mathbf{x}_i,\mathbf{x}_i)^2}\right]$$
$$+ R^2\sqrt{\frac{18}{m}\ln\left(\frac{2m}{\delta}\right)}$$

where the support of the distribution is in a ball of radius $R$ in the feature space and $\lambda_i$ and $\hat{\lambda}_i$ are the process and empirical eigenvalues respectively.

**Theorem 7** *If we perform PCA in the feature space defined by a kernel* $k(\mathbf{x}, \mathbf{z})$ *then with probability greater than* $1 - \delta$, *for all* $1 \leq k \leq m$, *if we project new data onto the space* $\hat{V}_k$, *the sum of the largest* $k$ *process eigenvalues is bounded by*

$$\lambda^{\leq k} \geq \mathbb{E}\left[\|P_{\hat{V}_k}(\psi(\mathbf{x}))\|^2\right] \geq \max_{1 \leq \ell \leq k}\left[\frac{1}{m}\hat{\lambda}^{\leq \ell}(S) - \frac{1+\sqrt{\ell}}{\sqrt{m}}\sqrt{\frac{2}{m}\sum_{i=1}^{m}k(\mathbf{x}_i,\mathbf{x}_i)^2}\right]$$
$$- R^2\sqrt{\frac{19}{m}\ln\left(\frac{2(m+1)}{\delta}\right)}$$

where the support of the distribution is in a ball of radius $R$ in the feature space and $\lambda_i$ and $\hat{\lambda}_i$ are the process and empirical eigenvalues respectively.

The proofs of these results are given in Shawe-Taylor et al. (2003). Theorem 6 implies that if $k \ll m$ the expected residual $\mathbb{E}\left[\|P_{\hat{V}_k}^{\perp}(\psi(\mathbf{x}))\|^2\right]$ closely matches the average sample residual of $\hat{\mathbb{E}}\left[\|P_{\hat{V}_k}^{\perp}(\psi(\mathbf{x}))\|^2\right] = (1/m)\sum_{i=k+1}^{m}\hat{\lambda}_i$, thus providing a bound for kernel PCA on new data. Theorem 7 implies a good fit between the partial sums of the largest $k$ empirical and process eigenvalues when $\sqrt{k/m}$ is small.

# References

Anderson, T. W. (1963). Asymptotic Theory for Principal Component Analysis. *Annals of Mathematical Statistics*, 34(1):122–148.

Baker, C. T. H. (1977). *The numerical treatment of integral equations*. Clarendon Press, Oxford.

Koltchinskii, V. and Gine, E. (2000). Random matrix approximation of spectra of integral operators. *Bernoulli*, 6(1):113–167.

Schölkopf, B., Smola, A., and Müller, K.-R. (1998). Nonlinear component analysis as a kernel eigenvalue problem. *Neural Computation*, 10:1299–1319.

Shawe-Taylor, J., Cristianini, N., and Kandola, J. (2002). On the Concentration of Spectral Properties. In Diettrich, T. G., Becker, S., and Ghahramani, Z., editors, *Advances in Neural Information Processing Systems 14*. MIT Press.

Shawe-Taylor, J., Williams, C. K. I., Cristianini, N., and Kandola, J. (2003). On the Eigenspectrum of the Gram Matrix and the Generalisation Error of Kernel PCA. Technical Report NC2-TR-2003-143, Dept of Computer Science, Royal Holloway, University of London. Available from `http://www.neurocolt.com/archive.html`.

Williams, C. K. I. and Seeger, M. (2000). The Effect of the Input Density Distribution on Kernel-based Classifiers. In Langley, P., editor, *Proceedings of the Seventeenth International Conference on Machine Learning (ICML 2000)*. Morgan Kaufmann.

Zhu, H., Williams, C. K. I., Rohwer, R. J., and Morciniec, M. (1998). Gaussian regression and optimal finite dimensional linear models. In Bishop, C. M., editor, *Neural Networks and Machine Learning*. Springer-Verlag, Berlin.
